# The Manifold Tangent Classifier

**Salah Rifai, Yann N. Dauphin, Pascal Vincent, Yoshua Bengio, Xavier Muller**
Department of Computer Science and Operations Research
University of Montreal
Montreal, H3C 3J7
{rifaisal, dauphiya, vincentp, bengioy, mullerx}@iro.umontreal.ca

## Abstract

We combine three important ideas present in previous work for building classifiers: the semi-supervised hypothesis (the input distribution contains information about the classifier), the unsupervised manifold hypothesis (data density concentrates near low-dimensional manifolds), and the manifold hypothesis for classification (different classes correspond to disjoint manifolds separated by low density). We exploit a novel algorithm for capturing manifold structure (high-order contractive auto-encoders) and we show how it builds a topological atlas of charts, each chart being characterized by the principal singular vectors of the Jacobian of a representation mapping. This representation learning algorithm can be stacked to yield a deep architecture, and we combine it with a domain knowledge-free version of the TangentProp algorithm to encourage the classifier to be insensitive to local directions changes along the manifold. Record-breaking classification results are obtained.

## 1 Introduction

Much of machine learning research can be viewed as an exploration of ways to compensate for scarce prior knowledge about how to solve a specific task by extracting (usually implicit) knowledge from vast amounts of data. This is especially true of the search for generic learning algorithms that are to perform well on a wide range of domains for which they were not specifically tailored. While such an outlook precludes using much domain-specific knowledge in designing the algorithms, it can however be beneficial to leverage what might be called "generic" prior hypotheses, that appear likely to hold for a wide range of problems. The approach studied in the present work exploits three such prior hypotheses:

1. The **semi-supervised learning hypothesis**, according to which learning aspects of the input distribution $p(x)$ can improve models of the conditional distribution of the supervised target $p(y|x)$, i.e., $p(x)$ and $p(y|x)$ share something (Lasserre *et al.*, 2006). This hypothesis underlies not only the strict semi-supervised setting where one has many more unlabeled examples at his disposal than labeled ones, but also the successful *unsupervised pretraining* approach for learning deep architectures, which has been shown to significantly improve supervised performance even without using *additional* unlabeled examples (Hinton *et al.*, 2006; Bengio, 2009; Erhan *et al.*, 2010).

2. The **(unsupervised) manifold hypothesis**, according to which real world data presented in high dimensional spaces is likely to concentrate in the vicinity of non-linear sub-manifolds of much lower dimensionality (Cayton, 2005; Narayanan and Mitter, 2010).

3. The **manifold hypothesis for classification**, according to which points of different classes are likely to concentrate along different sub-manifolds, separated by low density regions of the input space.

The recently proposed Contractive Auto-Encoder (CAE) algorithm (Rifai *et al.*, 2011a), based on the idea of encouraging the learned representation to be robust to small variations of the input, was shown to be very effective for unsupervised feature learning. Its successful application in the pre-training of deep neural networks is yet another illustration of what can be gained by adopting **hypothesis 1**. In addition, Rifai *et al.* (2011a) propose, and show empirical evidence for, the hypothesis that the trade-off between reconstruction error and the pressure to be insensitive to variations in input space has an interesting consequence: It yields a mostly contractive mapping that, locally around each training point, remains substantially sensitive only to a few input directions (with different directions of sensitivity for different training points). This is taken as evidence that the algorithm indirectly exploits **hypothesis 2** and models a lower-dimensional manifold. Most of the directions to which the representation is substantially sensitive are thought to be directions tangent to the data-supporting manifold (those that locally define its *tangent space*).

The present work follows through on this interpretation, and investigates whether it is possible to use this information, that is presumably captured about manifold structure, to further improve classification performance by leveraging **hypothesis 3**. To that end, we extract a set of basis vectors for the local tangent space at each training point from the Contractive Auto-Encoder's learned parameters. This is obtained with a Singular Value Decomposition (SVD) of the Jacobian of the encoder that maps each input to its learned representation. Based on hypothesis 3, we then adopt the "generic prior" that class labels are likely to be insensitive to most directions within these local tangent spaces (ex: small translations, rotations or scalings usually do not change an image's class). Supervised classification algorithms that have been devised to efficiently exploit tangent directions given as domain-specific prior-knowledge (Simard *et al.*, 1992, 1993), can readily be used instead with our learned tangent spaces. In particular, we will show record-breaking improvements by using TangentProp for fine tuning CAE-pre-trained deep neural networks. To the best of our knowledge this is the first time that the implicit relationship between an unsupervised learned mapping and the tangent space of a manifold is rendered explicit and successfully exploited for the training of a classifier. This showcases a unified approach that simultaneously leverages all three "generic" prior hypotheses considered. Our experiments (see Section 6) show that this approach sets new records for domain-knowledge-free performance on several real-world classification problems. Remarkably, in some cases it even outperformed methods that use weak or strong domain-specific prior knowledge (e.g. convolutional networks and tangent distance based on a-priori known transformations). Naturally, this approach is even more likely to be beneficial for datasets where no prior knowledge is readily available.

## 2 Contractive auto-encoders (CAE)

We consider the problem of the unsupervised learning of a non-linear feature extractor from a dataset $\mathcal{D} = \{x_1, \ldots, x_n\}$. Examples $x_i \in \mathbb{R}^d$ are i.i.d. samples from an unknown distribution $p(x)$.

### 2.1 Traditional auto-encoders

The auto-encoder framework is one of the oldest and simplest techniques for the unsupervised learning of non-linear feature extractors. It learns an *encoder* function $h$, that maps an input $x \in \mathbb{R}^d$ to a hidden representation $h(x) \in \mathbb{R}^{d_h}$, jointly with a *decoder* function $g$, that maps $h$ back to the input space as $r = g(h(x))$ the *reconstruction* of $x$. The encoder and decoder's parameters $\theta$ are learned by stochastic gradient descent to minimize the average *reconstruction error* $L(x, g(h(x)))$ for the examples of the training set. The objective being minimized is:

$$\mathcal{J}_{\text{AE}}(\theta) = \sum_{x \in \mathcal{D}} L(x, g(h(x))). \tag{1}$$

We will will use the most common forms of encoder, decoder, and reconstruction error:

**Encoder:** $h(x) = s(Wx + b_h)$, where $s$ is the element-wise logistic sigmoid $s(z) = \frac{1}{1+e^{-z}}$. Parameters are a $d_h \times d$ weight matrix $W$ and bias vector $b_h \in \mathbb{R}^{d_h}$.

**Decoder:** $r = g(h(x)) = s_2(W^T h(x) + b_r)$. Parameters are $W^T$ (tied weights, shared with the encoder) and bias vector $b_r \in \mathbb{R}^d$. Activation function $s_2$ is either a logistic sigmoid ($s_2 = s$) or the identity (linear decoder).

**Loss function:** Either the squared error: $L(x, r) = \|x - r\|^2$ or Bernoulli cross-entropy: $L(x, r) = -\sum_{i=1}^{d} x_i \log(r_i) + (1 - x_i) \log(1 - r_i)$.

The set of parameters of such an auto-encoder is $\theta = \{W, b_h, b_r\}$.

Historically, auto-encoders were primarily viewed as a technique for dimensionality reduction, where a narrow bottleneck (i.e. $d_h < d$) was in effect acting as a capacity control mechanism. By contrast, recent successes (Bengio *et al.*, 2007; Ranzato *et al.*, 2007a; Kavukcuoglu *et al.*, 2009; Vincent *et al.*, 2010; Rifai *et al.*, 2011a) tend to rely on rich, oftentimes over-complete representations ($d_h > d$), so that more sophisticated forms of regularization are required to pressure the auto-encoder to extract relevant features and avoid trivial solutions. Several successful techniques aim at sparse representations (Ranzato *et al.*, 2007a; Kavukcuoglu *et al.*, 2009; Goodfellow *et al.*, 2009). Alternatively, denoising auto-encoders (Vincent *et al.*, 2010) change the objective from mere reconstruction to that of denoising.

## 2.2 First order and higher order contractive auto-encoders

More recently, Rifai *et al.* (2011a) introduced the Contractive Auto-Encoder (CAE), that encourages robustness of representation $h(x)$ to small variations of a training input $x$, by penalizing its *sensitivity* to that input, measured as the Frobenius norm of the encoder's Jacobian $J(x) = \frac{\partial h}{\partial x}(x)$. The regularized objective minimized by the CAE is the following:

$$\mathcal{J}_{\text{CAE}}(\theta) = \sum_{x \in \mathcal{D}} L(x, g(h(x))) + \lambda \|J(x)\|^2, \tag{2}$$

where $\lambda$ is a non-negative regularization hyper-parameter that controls how strongly the norm of the Jacobian is penalized. Note that, with the traditional sigmoid encoder form given above, one can easily obtain the Jacobian of the encoder. Its $j^{th}$ row is obtained form the $j^{th}$ row of $W$ as:

$$J(x)_j = \frac{\partial h_j(x)}{\partial x} = h_j(x)(1 - h_j(x))W_j. \tag{3}$$

Computing the extra penalty term (and its contribution to the gradient) is similar to computing the reconstruction error term (and its contribution to the gradient), thus relatively cheap.

It is also possible to penalize higher order derivatives (Hessian) by using a simple stochastic technique that eschews computing them explicitly, which would be prohibitive. It suffices to penalize differences between the Jacobian at $x$ and the Jacobian at nearby points $\tilde{x} = x + \epsilon$ (stochastic corruptions of $x$). This yields the CAE+H (Rifai *et al.*, 2011b) variant with the following optimization objective:

$$\mathcal{J}_{\text{CAE+H}}(\theta) = \sum_{x \in \mathcal{D}} L(x, g(h(x))) + \lambda \|J(x)\|^2 + \gamma \mathbb{E}_{\epsilon \sim \mathcal{N}(0, \sigma^2 I)} \left[ \|J(x) - J(x + \epsilon)\|^2 \right], \tag{4}$$

where $\gamma$ is an additional regularization hyper-parameters that controls how strongly we penalize local variations of the Jacobian, i.e. higher order derivatives. The expectation $\mathbb{E}$ is over Gaussian noise variable $\epsilon$. In practice stochastic samples thereof are used for each stochastic gradient update. The CAE+H is the variant used for our experiments.

## 3 Characterizing the tangent bundle captured by a CAE

Rifai *et al.* (2011a) reason that, while the regularization term encourages insensitivity of $h(x)$ in all input space directions, this pressure is counterbalanced by the need for accurate reconstruction, thus resulting in $h(x)$ being substantially sensitive *only* to the few input directions required to distinguish close by training points. The geometric interpretation is that these directions span the local tangent space of the underlying manifold that supports the data. The *tangent bundle* of a smooth manifold is the manifold along with the set of tangent planes taken at all points on it. Each such tangent plane can be equipped with a local Euclidean coordinate system or *chart*. In topology, an *atlas* is a collection of such *charts* (like the locally Euclidean map in each page of a geographic atlas). Even though the set of charts may form a non-Euclidean manifold (e.g., a sphere), each chart is Euclidean.

## 3.1 Conditions for the feature mapping to define an atlas on a manifold

In order to obtain a proper atlas of charts, $h$ must be a diffeomorphism. It must be smooth ($C^\infty$) and invertible on open Euclidean balls on the manifold $\mathcal{M}$ around the training points. Smoothness is guaranteed because of our choice of parametrization (affine + sigmoid). Injectivity (different values of $h(x)$ correspond to different values of $x$) on the training examples is encouraged by minimizing reconstruction error (otherwise we cannot distinguish training examples $x_i$ and $x_j$ by only looking at $h(x_i)$ and $h(x_j)$). Since $h(x) = s(Wx + b_h)$ and $s$ is invertible, using the definition of injectivity we get (by composing $h(x_i) = h(x_j)$ with $s^{-1}$)

$$\forall i, j \ h(x_i) = h(x_j) \Longleftrightarrow W\Delta_{ij} = 0$$

where $\Delta_{ij} = x_i - x_j$. In order to preserve the injectivity of $h$, $W$ has to form a basis spanned by its rows $W_k$, where $\forall i, j \ \exists \alpha \in \mathbb{R}^{d_h}, \Delta_{ij} = \sum_k^{d_h} \alpha_k W_k$. With this condition satisfied, mapping $h$ is injective in the subspace spanned by the variations in the training set. If we limit the domain of $h$ to $h(\mathcal{X}) \subset (0,1)^{d_h}$ comprising values obtainable by $h$ applied to some set $\mathcal{X}$, then we obtain surjectivity by definition, hence *bijectivity of $h$ between the training set $\mathcal{D}$ and $h(\mathcal{D})$*. Let $\mathcal{M}_x$ be an open ball on the manifold $\mathcal{M}$ around training example $x$. By smoothness of the manifold $\mathcal{M}$ and of mapping $h$, we obtain bijectivity locally around the training examples (on the manifold) as well, i.e., between $\cup_{x \in \mathcal{D}} \mathcal{M}_x$ and $h(\cup_{x \in \mathcal{D}} \mathcal{M}_x)$.

## 3.2 Obtaining an atlas from the learned feature mapping

Now that we have necessary conditions for local invertibility of $h(x)$ for $x \in \mathcal{D}$, let us consider how to define the local chart around $x$ from the nature of $h$. Because $h$ must be sensitive to changes from an example $x_i$ to one of its neighbors $x_j$, but insensitive to other changes (because of the CAE penalty), we expect that this will be reflected in the spectrum of the Jacobian matrix $J(x) = \frac{\partial h(x)}{\partial x}$ at each training point $x$. In the ideal case where $J(x)$ has rank $k$, $h(x + \epsilon v)$ differs from $h(x)$ only if $v$ is in the span of the singular vectors of $J(x)$ with non-zero singular value. In practice, $J(x)$ has many tiny singular values. Hence, we define a local chart around $x$ using the Singular Value Decomposition of $J^T(x) = U(x)S(x)V^T(x)$ (where $U(x)$ and $V(x)$ are orthogonal and $S(x)$ is diagonal). The tangent plane $\mathcal{H}_x$ at $x$ is given by the span of the set of principal singular vectors $\mathcal{B}_x$:

$$\mathcal{B}_x = \{U_{\cdot k}(x) | S_{kk}(x) > \epsilon\} \quad \text{and} \quad \mathcal{H}_x = \{x + v | v \in \text{span}(\mathcal{B}_x)\},$$

where $U_{\cdot k}(x)$ is the $k$-th column of $U(x)$, and $\text{span}(\{z_k\}) = \{x | x = \sum_k w_k z_k, w_k \in \mathbb{R}\}$. We can thus define an atlas $\mathcal{A}$ captured by $h$, based on the local linear approximation around each example:

$$\mathcal{A} = \{(\mathcal{M}_x, \phi_x) | x \in \mathcal{D}, \phi_x(\tilde{x}) = \mathcal{B}_x(\tilde{x} - x)\}. \tag{5}$$

Note that this way of obtaining an atlas can also be applied to subsequent layers of a deep network. It is thus possible to use a greedy layer-wise strategy to initialize a network with CAEs (Rifai *et al.*, 2011a) and obtain an atlas that corresponds to the nonlinear features computed at any layer.

# 4 Exploiting the learned tangent directions for classification

Using the previously defined charts for every point of the training set, we propose to use this *additional information* provided by unsupervised learning to improve the performance of the supervised task. In this we adopt the **manifold hypothesis for classification** mentioned in the introduction.

## 4.1 CAE-based tangent distance

One way of achieving this is to use a nearest neighbor classifier with a similarity criterion defined as the shortest distance between two hyperplanes (Simard *et al.*, 1993). The tangents extracted on each points will allow us to shrink the distances between two samples when they can approximate each other by a linear combination of their local tangents. Following Simard *et al.* (1993), we define the *tangent distance* between two points $x$ and $y$ as the distance between the two hyperplanes $\mathcal{H}_x, \mathcal{H}_y \subset \mathbb{R}^d$ spanned respectively by $\mathcal{B}_x$ and $\mathcal{B}_y$. Using the usual definition of distance between two spaces, $d(\mathcal{H}_x, \mathcal{H}_y) = \inf\{\|z - w\|^2 | (z, w) \in \mathcal{H}_x \times \mathcal{H}_y\}$, we obtain the solution for this convex

problem by solving a system of linear equations (Simard *et al.*, 1993). This procedure corresponds to allowing the considered points $x$ and $y$ to move along the directions spanned by their associated local charts. Their distance is then evaluated on the new coordinates where the distance is minimal. We can then use a nearest neighbor classifier based on this distance.

## 4.2 CAE-based tangent propagation

Nearest neighbor techniques are often impractical for large scale datasets because their computational requirements scale linearly with $n$ for each test case. By contrast, once trained, neural networks yield fast responses for test cases. We can also leverage the extracted local charts when training a neural network. Following the *tangent propagation* approach of Simard *et al.* (1992), but exploiting our learned tangents, we encourage the output $o$ of a neural network classifier to be insensitive to variations in the directions of the local chart of $x$ by adding the following penalty to its supervised objective function:

$$\Omega(x) = \sum_{u \in \mathcal{B}_x} \left\| \frac{\partial o}{\partial x}(x)\, u \right\|^2 \tag{6}$$

Contribution of this term to the gradients of network parameters can be computed in $O(N_w)$, where $N_w$ is the number of neural network weights.

## 4.3 The Manifold Tangent Classifier (MTC)

Putting it all together, here is the high level summary of how we build and train a deep network:

1. Train (unsupervised) a stack of $K$ CAE+H layers (Eq. 4). Each is trained in turn on the representation learned by the previous layer.

2. For each $x_i \in \mathcal{D}$ compute the Jacobian of the last layer representation $J^{(K)}(x_i) = \frac{\partial h^{(K)}}{\partial x}(x_i)$ and its SVD[1]. Store the leading $d_M$ singular vectors in set $\mathcal{B}_{x_i}$.

3. On top of the $K$ pre-trained layers, stack an output layer of size the number of classes. Fine-tune the whole network for supervised classification[2] with an added tangent propagation penalty (Eq. 6), using for each $x_i$, tangent directions $\mathcal{B}_{x_i}$.

We call this deep learning algorithm the Manifold Tangent Classifier (MTC). Alternatively, instead of step 3, one can use the tangent vectors in $\mathcal{B}_{x_i}$ in a tangent distance nearest neighbors classifier.

# 5 Related prior work

Many **Non-Linear Manifold Learning** algorithms (Roweis and Saul, 2000; Tenenbaum *et al.*, 2000) have been proposed which can automatically discover the main directions of variation around each training point, i.e., the tangent bundle. Most of these algorithms are non-parametric and local, i.e., explicitly parametrizing the tangent plane around each training point (with a separate set of parameters for each, or derived mostly from the set of training examples in every neighborhood), as most explicitly seen in **Manifold Parzen Windows** (Vincent and Bengio, 2003) and manifold **Charting** (Brand, 2003). See Bengio and Monperrus (2005) for a critique of local non-parametric manifold algorithms: they might require a number of training examples which grows exponentially with manifold dimension and curvature (more crooks and valleys in the manifold will require more examples). One attempt to generalize the manifold shape non-locally (Bengio *et al.*, 2006) is based on explicitly predicting the tangent plane associated to any given point $x$, as a parametrized function of $x$. Note that these algorithms all explicitly exploit training set neighborhoods (see Figure 2), i.e. they use pairs or tuples of points, with the goal to explicitly model the tangent space, while it is

modeled implicitly by the CAE's objective function (that is *not* based on pairs of points). More recently, the **Local Coordinate Coding** (LCC) algorithm (Yu *et al.*, 2009) and its Local Tangent LCC variant (Yu and Zhang, 2010) were proposed to build a a local chart around each training example (with a local low-dimensional coordinate system around it) and use it to define a representation for each input $x$: the responsibility of each local chart/anchor in explaining input $x$ and the coordinate of $x$ in each local chart. That representation is then fed to a classifier and yield better generalization than $x$ itself.

The tangent distance (Simard *et al.*, 1993) and TangentProp (Simard *et al.*, 1992) algorithms were initially designed to exploit prior domain-knowledge of directions of invariance (ex: knowledge that the class of an image should be invariant to small translations rotations or scalings in the image plane). However any algorithm able to output a chart for a training point might potentially be used, as we do here, to provide directions to a Tangent distance or TangentProp (Simard *et al.*, 1992) based classifier. Our approach is nevertheless unique as the CAE's unsupervised feature learning capabilities are used *simultaneously* to provide a good initialization of deep network layers **and** a coherent non-local predictor of tangent spaces. TangentProp is itself closely related to the **Double Backpropagation** algorithm (Drucker and LeCun, 1992), in which one instead adds a penalty that is the sum of squared derivatives of the prediction error (with respect to the network input). Whereas TangentProp attempts to make the output insensitive to selected directions of change, the double backpropagation penalty term attempts to make the error at a training example invariant to changes in all directions. Since one is also trying to minimize the error at the training example, this amounts to making that minimization more robust, i.e., extend it to the *neighborhood* of the training examples.

Also related is the **Semi-Supervised Embedding** algorithm (Weston *et al.*, 2008). In addition to minimizing a supervised prediction error, it encourages each layer of representation of a deep architecture to be *invariant* when the training example is changed from $x$ to a near neighbor of $x$ in the training set. This algorithm works implicitly under the hypothesis that the variable $y$ to predict from $x$ is invariant to the local directions of change present between nearest neighbors. This is consistent with the **manifold hypothesis for classification** (hypothesis 3 mentioned in the introduction). Instead of removing variability along the local directions of variation, the **Contractive Auto-Encoder** (Rifai *et al.*, 2011a) *initially* finds a representation which is most *sensitive* to them, as we explained in section 2.

## 6   Experiments

We conducted experiments to evaluate our approach and the quality of the manifold tangents learned by the CAE, using a range of datasets from different domains:

**MNIST** is a dataset of $28 \times 28$ images of handwritten digits. The learning task is to predict the digit contained in the images. **Reuters Corpus Volume I** is a popular benchmark for document classification. It consists of 800,000 real-world news wire stories made available by Reuters. We used the 2000 most frequent words calculated on the whole dataset to create a bag-of-words vector representation. We used the LYRL2004 split to separate between a train and test set. **CIFAR-10** is a dataset of 70,000 $32 \times 32$ RGB real-world images. It contains images of real-world objects (i.e. cars, animals) with all the variations present in natural images (i.e. backgrounds). **Forest Cover Type** is a large-scale database of cartographic variables for the prediction of forest cover types made available by the US Forest Service.

We investigate whether leveraging the CAE learned tangents leads to better classification performance on these problems, using the following methodology: Optimal hyper-parameters for (a stack of) CAEs are selected by cross-validation on a disjoint validation set extracted from the training set. The quality of the feature extractor and tangents captured by the CAEs is evaluated by initializing an neural network (MLP) with the same parameters and fine-tuning it by backpropagation on the supervised classification task. The optimal strength of the supervised TangentProp penalty and number of tangents $d_M$ is also cross-validated.

**Results**

Figure 1 shows a visualization of the tangents learned by the CAE. On MNIST, the tangents mostly correspond to small geometrical transformations like translations and rotations. On CIFAR-10, the

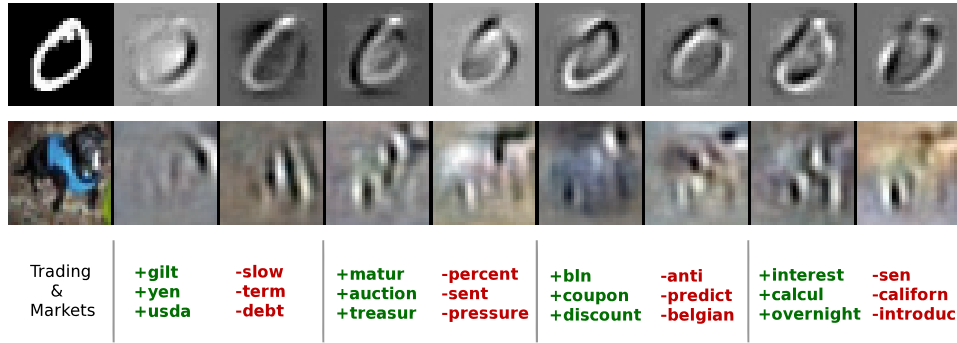

| Trading & Markets | +gilt +yen +usda | -slow -term -debt | +matur +auction +treasur | -percent -sent -pressure | +bln +coupon +discount | -anti -predict -belgian | +interest +calcul +overnight | -sen -californ -introduc |

Figure 1: Visualisation of the tangents learned by the CAE for MNIST, CIFAR-10 and RCV1 (top to bottom). The left-most column is the example and the following columns are its tangents. On RCV1, we show the tangents of a document with the topic "Trading & Markets" (MCAT) with the negative terms in red(-) and the positive terms in green(+).

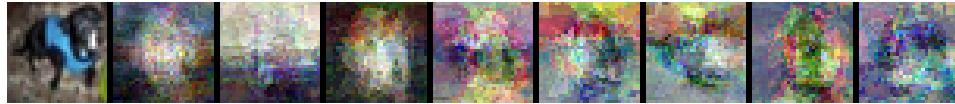

Figure 2: Tangents extracted by local PCA on CIFAR-10. This shows the limitation of approaches that rely on training set neighborhoods.

model also learns sensible tangents, which seem to correspond to changes in the *parts* of objects. The tangents on RCV1-v2 correspond to the addition or removal of similar words and removal of irrelevant words. We also note that extracting the tangents of the model is a way to visualize what the model has learned about the structure of the manifold. Interestingly, we see that **hypothesis 3** holds for these datasets because most tangents do not change the class of the example.

Table 1: Classification accuracy on several datasets using KNN variants measured on 10,000 test examples with 1,000 training examples. The KNN is trained on the raw input vector using the Euclidean distance while the K-layer+KNN is computed on the representation learned by a K-layer CAE. The KNN+Tangents uses at every sample the local charts extracted from the 1-layer CAE to compute tangent distance.

|  | KNN | KNN+Tangents | 1-Layer CAE+KNN | 2-Layer CAE+KNN |
|---|---|---|---|---|
| MNIST | 86.9 | 88.7 | 90.55 | **91.15** |
| CIFAR-10 | 25.4 | **26.5** | 25.1 | - |
| COVERTYPE | 70.2 | **70.98** | 69.54 | 67.45 |

We use KNN using tangent distance to evaluate the quality of the learned tangents more objectively. Table 1 shows that using the tangents extracted from a CAE always lead to better performance than a traditional KNN.

As described in section 4.2, the tangents extracted by the CAE can be used for fine-tuning the multi-layer perceptron using tangent propagation, yielding our Manifold Tangent Classifier (MTC). As it is a semi-supervised approach, we evaluate its effectiveness with a varying amount of labeled examples on MNIST. Following Weston *et al.* (2008), the unsupervised feature extractor is trained on the full training set and the supervised classifier is trained on a restricted labeled set. Table 2 shows our results for a single hidden layer MLP initialized with CAE+H pretraining (noted CAE for brevity) and for the same classifier fine-tuned with tangent propagation (i.e. the *manifold tangent classifier* of section 4.3, noted MTC). The methods that do not leverage the semi-supervised learning hypothesis (Support Vector Machines, traditional Neural Networks and Convolutional Neural Networks) give very poor performance when the amount of labeled data is low. In some cases, the methods that can learn from unlabeled data can reduce the classification error by half. The CAE gives better results than other approaches across almost the whole range considered. It shows that the features extracted

Table 2: Semi-supervised classification error on the MNIST test set with 100, 600, 1000 and 3000 labeled training examples. We compare our method with results from (Weston *et al.*, 2008; Ranzato *et al.*, 2007b; Salakhutdinov and Hinton, 2007).

|      | NN    | SVM   | CNN   | TSVM  | DBN-rNCA | EmbedNN | CAE   | MTC      |
|------|-------|-------|-------|-------|----------|---------|-------|----------|
| 100  | 25.81 | 23.44 | 22.98 | 16.81 | -        | 16.86   | 13.47 | **12.03** |
| 600  | 11.44 | 8.85  | 7.68  | 6.16  | 8.7      | 5.97    | 6.3   | **5.13**  |
| 1000 | 10.7  | 7.77  | 6.45  | 5.38  | -        | 5.73    | 4.77  | **3.64**  |
| 3000 | 6.04  | 4.21  | 3.35  | 3.45  | 3.3      | 3.59    | 3.22  | **2.57**  |

from the rich unlabeled data distribution give a good inductive prior for the classification task. Note that the MTC consistently outperforms the CAE on this benchmark.

Table 3: Classification error on the MNIST test set with the full training set.

| K-NN  | NN    | SVM   | DBN   | CAE   | DBM   | CNN   | MTC     |
|-------|-------|-------|-------|-------|-------|-------|---------|
| 3.09% | 1.60% | 1.40% | 1.17% | 1.04% | 0.95% | 0.95% | **0.81**% |

Table 3 shows our results on the full MNIST dataset with some results taken from (LeCun *et al.*, 1999; Hinton *et al.*, 2006). The CAE in this figure is a two-layer deep network with 2000 units per layer pretrained with the CAE+H objective. The MTC uses the same stack of CAEs trained with tangent propagation using 15 tangents. The prior state of the art for the permutation invariant version of the task was set by the Deep Boltzmann Machines (Salakhutdinov and Hinton, 2009) at 0.95%. Using our approach, we reach 0.81% error on the test set. Remarkably, the MTC also outperforms the basic Convolutional Neural Network (CNN) even though the CNN exploits prior knowledge about vision using convolution and pooling to enhance the results.

Table 4: Classification error on the Forest CoverType dataset.

| SVM   | Distributed SVM | MTC     |
|-------|-----------------|---------|
| 4.11% | 3.46%           | **3.13**% |

We also trained a 4 layer MTC on the Forest CoverType dataset. Following Trebar and Steele (2008), we use the data split DS2-581 which contains over 500,000 training examples. The MTC yields the best performance for the classification task beating the previous state of the art held by the distributed SVM (mixture of several non-linear SVMs).

# 7 Conclusion

In this work, we have shown a new way to characterize a manifold by extracting a local chart at each data point based on the unsupervised feature mapping built with a deep learning approach. The developed Manifold Tangent Classifier successfully leverages three common "generic prior hypotheses" in a unified manner. It learns a meaningful representation that captures the structure of the manifold, and can leverage this knowledge to reach superior classification performance. On datasets from different domains, it successfully achieves state of the art performance.

**Acknowledgments** The authors would like to acknowledge the support of the following agencies for research funding and computing support: NSERC, FQRNT, Calcul Québec and CIFAR.

## Footnotes

[1] $J^{(K)}$ is the product of the Jacobians of each encoder (see Eq. 3) in the stack. It suffices to compute its leading $d_M$ SVD vectors and singular values. This is achieved in $O(d_M \times d \times d_h)$ per training example. For comparison, the cost of a forward propagation through a single MLP layer is $O(d \times d_h)$ per example.

[2] A sigmoid output layer is preferred because computing its Jacobian is straightforward and efficient (Eq. 3). The supervised cost used is the cross entropy. Training is by stochastic gradient descent.

# References

Bengio, Y. (2009). Learning deep architectures for AI. *Foundations and Trends in Machine Learning*, **2**(1), 1–127. Also published as a book. Now Publishers, 2009.

Bengio, Y. and Monperrus, M. (2005). Non-local manifold tangent learning. In *NIPS'04*, pages 129–136. MIT Press.

Bengio, Y., Larochelle, H., and Vincent, P. (2006). Non-local manifold parzen windows. In *NIPS'05*, pages 115–122. MIT Press.

Bengio, Y., Lamblin, P., Popovici, D., and Larochelle, H. (2007). Greedy layer-wise training of deep networks. In *Advances in NIPS 19*.

Brand, M. (2003). Charting a manifold. In *NIPS'02*, pages 961–968. MIT Press.

Cayton, L. (2005). Algorithms for manifold learning. Technical Report CS2008-0923, UCSD.

Drucker, H. and LeCun, Y. (1992). Improving generalisation performance using double back-propagation. *IEEE Transactions on Neural Networks*, **3**(6), 991–997.

Erhan, D., Bengio, Y., Courville, A., Manzagol, P.-A., Vincent, P., and Bengio, S. (2010). Why does unsupervised pre-training help deep learning? *JMLR*, **11**, 625–660.

Goodfellow, I., Le, Q., Saxe, A., and Ng, A. (2009). Measuring invariances in deep networks. In *NIPS'09*, pages 646–654.

Hinton, G. E., Osindero, S., and Teh, Y. (2006). A fast learning algorithm for deep belief nets. *Neural Computation*, **18**, 1527–1554.

Kavukcuoglu, K., Ranzato, M., Fergus, R., and LeCun, Y. (2009). Learning invariant features through topographic filter maps. pages 1605–1612. IEEE.

Lasserre, J. A., Bishop, C. M., and Minka, T. P. (2006). Principled hybrids of generative and discriminative models. pages 87–94, Washington, DC, USA. IEEE Computer Society.

LeCun, Y., Haffner, P., Bottou, L., and Bengio, Y. (1999). Object recognition with gradient-based learning. In *Shape, Contour and Grouping in Computer Vision*, pages 319–345. Springer.

Narayanan, H. and Mitter, S. (2010). Sample complexity of testing the manifold hypothesis. In J. Lafferty, C. K. I. Williams, J. Shawe-Taylor, R. Zemel, and A. Culotta, editors, *Advances in Neural Information Processing Systems 23*, pages 1786–1794.

Ranzato, M., Poultney, C., Chopra, S., and LeCun, Y. (2007a). Efficient learning of sparse representations with an energy-based model. In *NIPS'06*.

Ranzato, M., Huang, F., Boureau, Y., and LeCun, Y. (2007b). Unsupervised learning of invariant feature hierarchies with applications to object recognition. IEEE Press.

Rifai, S., Vincent, P., Muller, X., Glorot, X., and Bengio, Y. (2011a). Contracting auto-encoders: Explicit invariance during feature extraction. In *Proceedings of the Twenty-eight International Conference on Machine Learning (ICML'11)*.

Rifai, S., Mesnil, G., Vincent, P., Muller, X., Bengio, Y., Dauphin, Y., and Glorot, X. (2011b). Higher order contractive auto-encoder. In *European Conference on Machine Learning and Principles and Practice of Knowledge Discovery in Databases (ECML PKDD)*.

Roweis, S. and Saul, L. K. (2000). Nonlinear dimensionality reduction by locally linear embedding. *Science*, **290**(5500), 2323–2326.

Salakhutdinov, R. and Hinton, G. E. (2007). Learning a nonlinear embedding by preserving class neighbourhood structure. In *AISTATS'2007*, San Juan, Porto Rico. Omnipress.

Salakhutdinov, R. and Hinton, G. E. (2009). Deep Boltzmann machines. In *AISTATS'2009*, volume 5, pages 448–455.

Simard, P., Victorri, B., LeCun, Y., and Denker, J. (1992). Tangent prop - A formalism for specifying selected invariances in an adaptive network. In *NIPS'91*, pages 895–903, San Mateo, CA. Morgan Kaufmann.

Simard, P. Y., LeCun, Y., and Denker, J. (1993). Efficient pattern recognition using a new transformation distance. In *NIPS'92*, pages 50–58. Morgan Kaufmann, San Mateo.

Tenenbaum, J., de Silva, V., and Langford, J. C. (2000). A global geometric framework for nonlinear dimensionality reduction. *Science*, **290**(5500), 2319–2323.

Trebar, M. and Steele, N. (2008). Application of distributed svm architectures in classifying forest data cover types. *Computers and Electronics in Agriculture*, **63**(2), 119 – 130.

Vincent, P. and Bengio, Y. (2003). Manifold parzen windows. In *NIPS'02*. MIT Press.

Vincent, P., Larochelle, H., Lajoie, I., Bengio, Y., and Manzagol, P.-A. (2010). Stacked denoising autoencoders: Learning useful representations in a deep network with a local denoising criterion. *JMLR*, **11**(3371–3408).

Weston, J., Ratle, F., and Collobert, R. (2008). Deep learning via semi-supervised embedding. In *ICML 2008*, pages 1168–1175, New York, NY, USA.

Yu, K. and Zhang, T. (2010). Improved local coordinate coding using local tangents.

Yu, K., Zhang, T., and Gong, Y. (2009). Nonlinear learning using local coordinate coding. In Y. Bengio, D. Schuurmans, J. Lafferty, C. K. I. Williams, and A. Culotta, editors, *Advances in Neural Information Processing Systems 22*, pages 2223–2231.

